# Adaptive Stimulus Representations:
# A Computational Theory of
# Hippocampal-Region Function

**Mark A. Gluck**        **Catherine E. Myers**
Center for Molecular and Behavioral Neuroscience
Rutgers University, Newark, NJ 07102
*gluck@pavlov.rutgers.edu    myers@pavlov.rutgers.edu*

## Abstract

We present a theory of cortico-hippocampal interaction in discrimination learning. The hippocampal region is presumed to form new stimulus representations which facilitate learning by enhancing the discriminability of predictive stimuli and compressing stimulus-stimulus redundancies. The cortical and cerebellar regions, which are the sites of long-term memory, may acquire these new representations but are not assumed to be capable of forming new representations themselves. Instantiated as a connectionist model, this theory accounts for a wide range of trial-level classical conditioning phenomena in normal (intact) and hippocampal-lesioned animals. It also makes several novel predictions which remain to be investigated empirically. The theory implies that the hippocampal region is involved in even the simplest learning tasks; although hippocampal-lesioned animals may be able to use other strategies to learn these tasks, the theory predicts that they will show consistently different patterns of transfer and generalization when the task demands change.

## 1 INTRODUCTION

It has long been known that the hippocampal region (including the entorhinal cortex, subicular complex, hippocampus and dentate gyrus) plays a role in learning and memory. For example, the hippocampus has been implicated in human declarative memory (Scoville & Millner, 1957; Squire, 1987) while hippocampal damage in animals impairs such seemingly disparate abilities as spatial mapping (O'Keefe & Nadel, 1978), contextual sensitivity (Hirsh, 1974; Winocur, Rawlins & Gray, 1987; Nadel & Willner, 1980), temporal processing (Buszaki, 1989; Akase, Alkon & Disterhoft, 1989), configural association (Sutherland & Rudy, 1989) and the flexible use of representations in novel situations (Eichenbaum & Buckingham, 1991). Several theories have characterized hippocampal function in terms of one or more of these abilities. However, a theory which can predict the full range of deficits after hippocampal lesion has been elusive.

This paper attempts to provide a functional interpretation of a hippocampal-region role in associative learning. We propose that one function of the hippocampal region is to construct new representations which facilitate discrimination learning. We argue that this

representational function is sufficient to derive and unify a wide range of trial-level conditioned effects observable in the intact and lesioned animal.

## 2  A THEORY OF CORTICO-HIPPOCAMPAL INTERACTION

Psychological theories have often found it useful to characterize stimuli as occupying points in an internal representation space (c.f. Shepard, 1958; Nosofsky, 1984). Connectionist theories can be interpreted in a similar geometric framework. For example, in a connectionist network (see Figure 1A) a stimulus input such as a tone is recoded in the network's internal layer as a pattern of activations. A light input will activate a different pattern of activations in the internal layer nodes (Figure 1B). These internal layer activations can be viewed as a representation of the stimulus inputs, and can be plotted in multi-dimensional internal representation space (Figure 1C). Learning to classify stimulus inputs corresponds to finding an appropriate partition of representation space. In the connectionist model, the lower layer of network weights determine the representation while the upper layer of network weights determine the classification.

Our basic premise is that the hippocampal region has the ability to modify stimulus representations to facilitate classification, and that its representations are biased by two constraints. The first constraint, predictive differentiation, is a bias to differentiate the representations of stimuli which are to be classified differently. Predictive differentiation increases the representational resources (i.e., hidden units) devoted to representing stimulus features which are especially predictive of how a stimulus is to be classified. For example, if red stimuli alone should evoke a response, then many representational

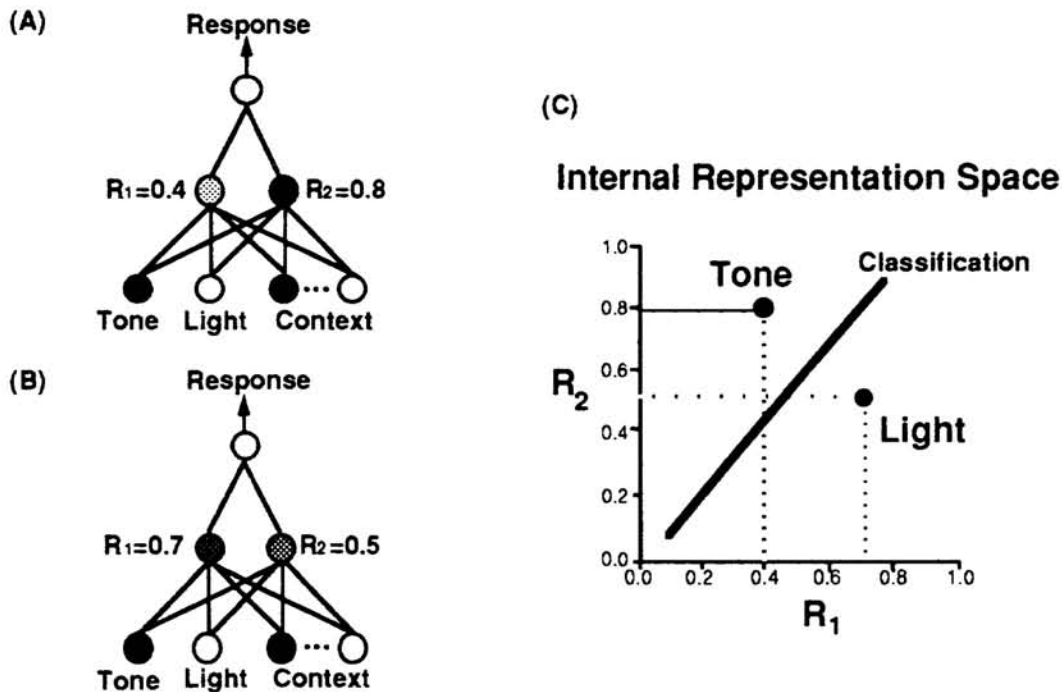

Figure 1: Stimulus representations. The activations of the internal layer nodes in a connectionist network constitute a representation of the network's stimulus inputs. (A) Internal representation for an example tone stimulus. (B) Internal representation for an example light stimulus. (C) Translation of these representations into points in an internal representation space, with one dimension encoding the activation level of each internal node. Classifying stimuli corresponds to partitioning representation space so that representations of stimuli which ought to be classified together lie in the same partition. Classification is easier if the representations of stimuli to be classified together are clustered while representations of stimuli to be classified differently are widely separated in this space.

resources should be devoted to encoding color. The second constraint, redundancy compression, reduces the resources allocated to represent features which are redundant or irrelevant in predicting the desired response. These two constraints are by nature complementary, given a finite amount of representational resources. Compressing redundant features frees resources to encode more predictive features. Conversely, increasing the resources allocated to predictive features forces compression of the remaining (less predictive) features.

This proposed hippocampal-region function may be modelled by a predictive autoencoder (on the right in Figure 2). An autoencoder (Hinton, 1989) learns to map from stimulus inputs, through an internal layer, to an output which is a reproduction of those inputs. This is also known as **stimulus-stimulus learning**. To do this, the network must have access to some multi-layer learning algorithm such as error backpropagation (Rumelhart, Hinton & Williams, 1986). When the internal layer is narrower than the input and output layers, the system develops a recoding in the internal layer which takes advantage of redundancies in the inputs. A predictive autoencoder has the further constraint that it must also output a classification response to the inputs. This is also known as **stimulus-response learning**. The internal layer recoding must therefore also emphasize stimulus features which are especially predictive of this classification. Therefore, a predictive autoencoder learns to form internal representations constrained by both predictive differentiation and redundancy compression, and is thus an example of a mechanism for implementing the two representational biases described above.

The cerebral and cerebellar cortices form the sites of long term memory in this theory, but are not themselves directly able to form new representations. They can, however, acquire new representations formed in the hippocampal region. A simplified model of one such cerebellar region is shown on the left in Figure 2. This network does not have access to multi-layer learning which would allow it to independently form new internal representations by itself. Instead, the two layers of weights in this network evolve independently. The bottom layer of weights is trained so that the current input pattern generates an internal representation equivalent to that developed in the hippocampal model. Independently and simultaneously, weights in the cortical network top layer are trained to map from this evolving representation to the classification response. Because the cortical networks are not creating new representations, but only learning two independent single-layer mappings, they can use a much simpler learning rule than the hippocampal model. One such algorithm is the LMS learning rule (Widrow & Hoff, 1960), which can instantiate the Rescorla-Wagner (1972) model of classical conditioning.

## Cortical (Cerebellar) Network    Hippocampal-System Model

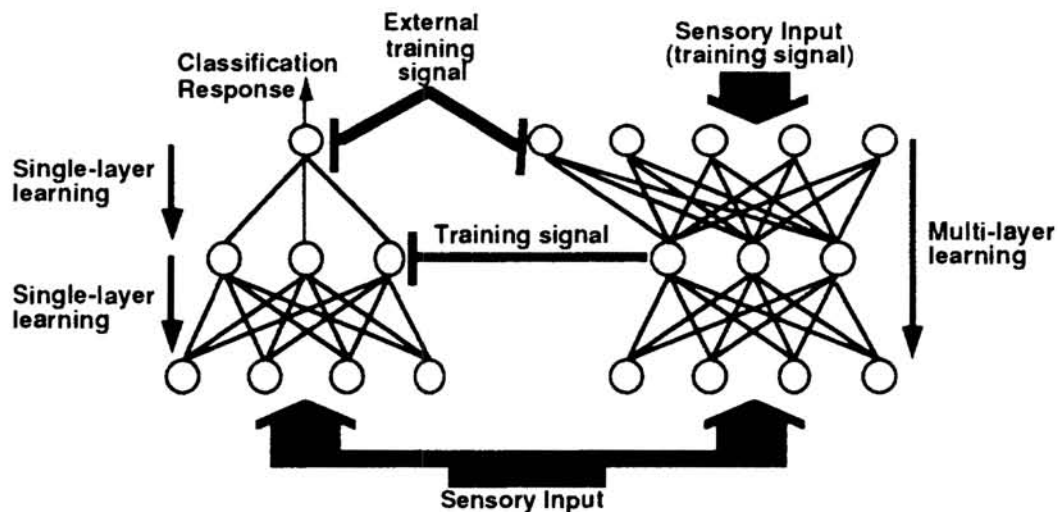

Figure 2. The cortico-hippocampal model: new representations developed in the hippocampal model can be acquired by cortical networks which are incapable of developing such representations by themselves.

# 3 MODELLING HIPPOCAMPAL INVOLVEMENT IN CLASSICAL CONDITIONING

A popular experimental paradigm for the study of associative learning in animals is classical conditioning of the rabbit eyeblink response (see Gormezano, Kehoe & Marshall, 1983, for review). A puff of air delivered to the eye elicits a blink response in the rabbit. If a previously neutral stimulus, such as a tone or light (called the **conditioned stimulus**), is repeatedly presented just before the airpuff, the animal will develop a blink response to this stimulus -- and time the response so that the lid is maximally closed just when the airpuff is scheduled to arrive. Ignoring the many temporal factors -- such as the interval between stimuli or precise timing of the response -- this reduces to a classification problem: learning which stimuli accurately predict the airpuff and should therefore evoke a response.

During a training trial, both the hippocampal and cortical networks receive the same input pattern. This pattern represents the presence or absence of all stimulus cues -- both conditioned stimuli and background contextual cues. Contextual cues are always present, but may change slowly over time. The hippocampus is trained incrementally to predict the current values of all cues -- including the US. The evolving hippoocampal internal layer representation is provided to the cortical network, which concurrently learns to reproduce this representation and to associate this evolving internal representation with a prediction of the US. This cortical network prediction is interpreted as the system's response.

The complete (intact) cortico-hippocampal model of Figure 2 can be shown to produce conditioned behavior comparable to that of normal (intact) animals. Hippocampal lesions can be simulated by disabling the hippocampal model. This eliminates the training signal which the cortical model would otherwise use to construct internal layer representations. As a result, the lower layer of cortical network weights remains fixed. The lesioned model's cortical network can still modify its upper layer of weights to learn new discriminations for which its current (now fixed) internal representation is sufficient.

# 4 BEHAVIORAL RESULTS

A stimulus discrimination task involves learning that one stimulus A predicts the airpuff but a second stimulus B does not. The notation <A+, B-> is used to indicate a series of training trials intermixing A+ (A preceeds the airpuff), B- (B does not preceed the airpuff) and context-alone presentations. Figure 3A shows the appropriate development of responses to A but not to B during this task. Both the intact and lesioned systems can acquire this discrimination. In fact, the lesioned system learns somewhat faster: it is only learning a classification, since its representation is fixed and (for this simple task) generally sufficient. In the intact system, by contrast, the hippocampal model is developing a new representation and transferring it to the cortical network The cortical network must then learn classifications based on this changing representation. This will be slower than learning based on a fixed representation. This paradox of discrimination facilitation after hippocampal lesion has often been reported in the animal literature (Schmaltz & Theios, 1972; Eichenbaum, Fagan, Mathews & Cohen, 1988); one previous interpretation has been to suggest that the hippocampal region is somehow "unneccessary for" or even "inhibitory to" simple discrimination learning. Our model suggests a different interpretation: the intact system learns more slowly because it is actually learning more than the lesioned system. The intact system is learning not only how to map from stimuli to responses, it is also developing new stimulus representations which enhance the differentiation among representations of predictive stimulus features while compressing the representations of redundant and irrelevant stimulus features.

The benefit of this re-representation can most readily be seen when the task demands suddenly change. For example, suppose the task valences shift from <A+, B-> to <A-, B+>. The representation developed during the first training phase, which maximally differentiated features distinguishing stimulus A from B, will still be useful in the second training phase. Only the classification needs to be relearned. Figure 3B shows that the intact system can learn the reversed task slightly more quickly than it learned the original task. Successive reversals are expected to be even more facilitated, as the representations of A and B grow ever more distinct (see Sutherland & Mackintosh, 1971, for a review of the relevant empirical data). In contrast, the lesioned system is severely impaired in the reversal task (Figure 3B). In the lesioned system, with a fixed representation, all the information is contained in the upper classificatory layer of weights. This information must be unlearned before the reversal task can be learned. Consistent with the model's behavior, empirical studies of hippocampal-lesioned animals show strong impairment at reversal learning (Berger & Orr, 1983).

The simplest evidence for redundancy compression likewise occurs during a transfer task. During unreinforced pre-exposure to a stimulus cue A, the presence or absence of A is irrelevant in terms of predicting US arrival (since a US never comes). Our theory expects that the representation of A will therefore become compressed with the representations of of the background contextual cues. In a subsequent training phase in which A does predict the US, the system must learn to respond to a feature it previously learned to ignore. The representation of A must now be re-differentiated from the context. Our theory therefore expects that learning to respond to A will be slowed, relative to learning

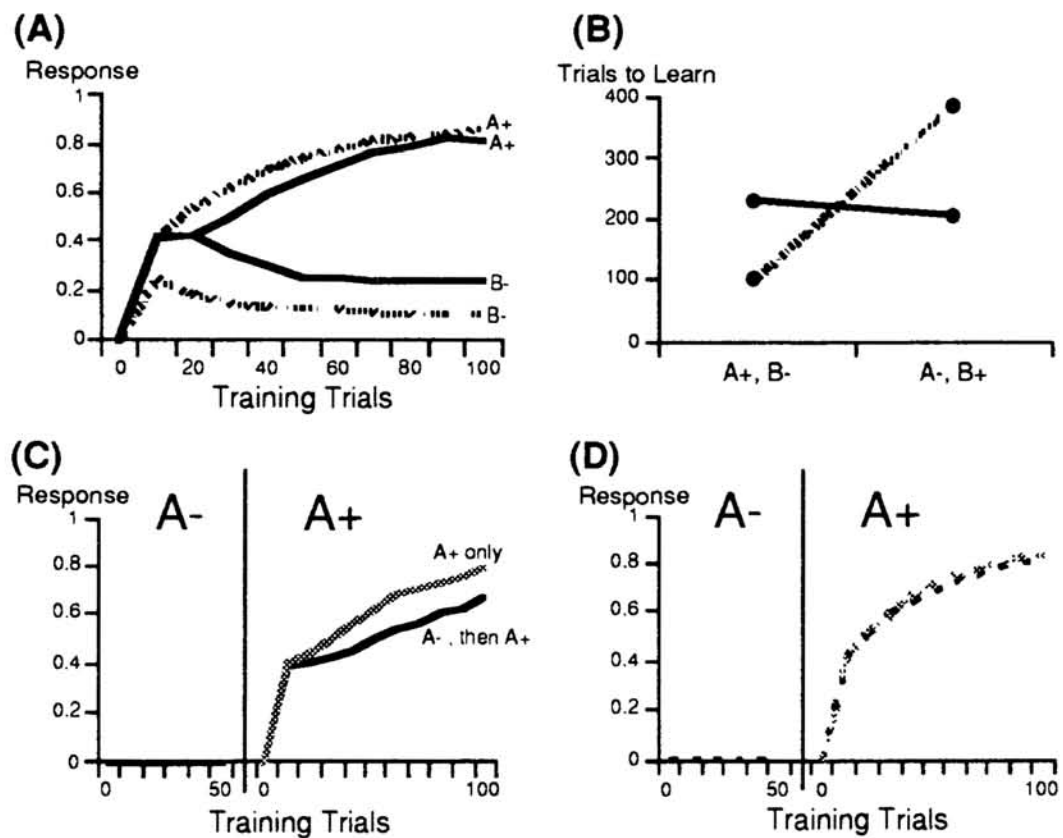

Figure 3. Behavioral results. Solid line = intact system, dashed line = lesioned system (A) Discrimination learning <A+, B-> in intact and lesioned models; lesioned model learns slightly faster. (B) Discrimination reversal (<A+, B-> then <A-, B+>) Intact system shows facilitation on successive reversals, lesioned system is severely impaired. (C) Latent inhibition (A- impairs A+) in the intact model; (D) No latent inhibition in the lesioned model. All results shown are consistent with empirical data (see text for references).

without pre-exposure to A (Figure 3C). This effect occurs in animals and is known as latent inhibition (Lubow, 1973).

In this theory, latent inhibition arises from hippocampal-dependent recodings. In the lesioned system, there is no stimulus-stimulus learning during the pre-exposure phase, and no redundancy compression in the (fixed) internal representation. Therefore, unreinforced pre-exposure does not slow the learning of a response to A (Figure 3D). Empirical studies have shown that hippocampal lesions also eliminate latent inhibition in animals (Solomon & Moore, 1975).

Incidentally, a standard feedforward backpropagation network, with the same architecture as the cortical network, but with access to a multi-layer learning algorithm, fails to show latent inhibition. Such a network can form representations in its internal layer, but unlike the hippocampal model it does not perform stimulus-stimulus learning. Therefore, there is no effect of unreinforced pre-exposure of a stimulus, and no latent inhibition effect (simulations not shown).

This cortico-hippocampal theory can account for many other effects of hippocampal lesions (see Gluck & Myers, 1992, 1993 / in press): including increased stimulus generalization and elimination of sensory preconditioning. It also provides an interpretation of the observation that hippocampal disruption can damage learning more than complete hippocampal removal (Solomon, Solomon, van der Schaaf & Perry, 1983): if the training signals from the hippocampus are "noisy", the cortical network will acquire a distorted and continuously changing internal representation. In general, this will make classification learning harder than in the lesioned system where the internal representation is simply fixed.

The theory also makes several novel and testable predictions. For example, in the intact animal, training to discriminate two highly similar stimuli is facilitated by pre-training on an easier version of the same task -- even if the hard task is a reversal of the easy task (Mackintosh & Little, 1970). The theory predicts that this effect arises from predictive differentiation during the pre-training phase, and therefore should be eliminated after hippocampal lesion. Another effect observed in intact animals is compound preconditioning: discrimination of two stimuli A and B is impaired by pre-exposure to the compound AB (Lubow, Rifkin & Alek, 1976). The theory attributes this effect to redundancy compression in the pre-exposure phase, and therefore again predicts that the effect should disappear in the hippocampal-lesioned animal.

## 5 CONCLUSIONS

There are many hippocampal-dependent phenomena which the model, in its present form, does not address. For example, the model does not consider real-time temporal effects, or operant choice behavior. Because it is a trial-level model, it does not address the issue of a consolidation period during which memories gradually become independent of the hippocampus. We have also not considered here the physiological mechanisms or structures within the hippocampal region which might implement the proposed hippocampal function. Finally, the model would require extensions before it could apply to such high-level behaviors as spatial navigation, human declarative memory, and working memory -- all of which are known to be disrupted by hippocampal lesions.

Despite the theory's restricted scope, it provides a simple and unified account of a wide range of trial-level conditioning data. It also makes several novel predictions which remain to be investigated in lesioned animals. The theory suggests that the effects of hippocampal damage may be especially informative in studies of two-phase transfer tasks. In these paradigms, both intact and hippocampal-lesioned animals are expected to behave similarly on a simple initial learning task, but exhibit different behaviors on a subsequent transfer or generalization task.

## REFERENCES

Akase, E., Alkon, D., & Disterhoft, J. (1989). Hippocampal lesions impair memory of short-delay conditioned eye blink in rabbits. Behavioral Neuroscience, 103(5), 935-943.

Berger, T. W., & Orr, W. B. (1983). Hippocampectomy selectively disrupts discrimination reversal learning of the rabbit nictitating membrane response. Behavioral Brain Research, 8, 49-68.

Buszaki, G. (1989). Two-stage model of memory trace formation: A role for "noisy" brain states. Neuroscience, 31(3), 551-570.

Eichenbaum, H., & Buckingham, J. (1991). Studies on hippocampal processing: Experiment, theory, and model. In M. Gabriel & J. Moore (Eds.), Neurocomputation and learning: Foundations of adaptive networks Cambridge, MA: M.I.T. Press.

Eichenbaum, H., Fagan, A., Mathews, P., & Cohen, N. (1988). Hippocampal system dysfunction and odor discrimination learning in rats: Impairment or facilitation depending on representational demands. Behavioral Neuroscience, 102(3), 331-339.

Gluck, M. & Myers, C. (1992). Hippocampal-system function in stimulus representation and generalization: A computational theory. Proceedings 14th Annual Conference of the Cognitive Science Society, Bloomington, IN, 390-395.

Gluck, M., & Myers, C. (1993 / in press). Hippocampal mediation of stimulus representation: A computational theory. Hippocampus.

Gormezano, I., Kehoe, E. K., & Marshal, B. S. (1983). Twenty years of classical conditioning research with the rabbit. Progress in Psychobiology and Physiological Psychology, 10, 197-275.

Hinton, G. E. (1989). Connectionist learning procedures. Artificial Intelligence, 40, 185-234.

Hirsh, R. (1974). The hippocampus and contextual retrieval of information from memory: A theory. Behavioral Biology, 12, 421-444.

Lubow, R. E. (1973). Latent inhibition. Psychological Bulletin, 79, 398-407.

Lubow, R., Rifkin, B., & Alek, M. (1976). The context effect: The relationship between stimulus pre-exposure and environmental pre-exposure determines subsequent learning. Journal of Experimental Psychology: Animal Behavior Processes, 2(1), 38-47.

Mackintosh, N. & Little, L. (1970). An analysis of transfer along a continuum. Canad. J. Psychol. / Rev. Canad. Psychol., 24(5), 362-369.

Nadel, L., & Willner, J. (1980). Context and conditioning: A place for space. Physiological Psychology, 8, 218-228.

Nosofsky, R. M. (1974). Choice, similarity, and the context theory of classification. Journal of Experimental Psychology: Learning, Memory and Cognition, 10, 104-114.

O'Keefe, J., & Nadel, L. (1978). The Hippocampus as a Cognitive Map. Oxford, UK: Claredon University Press.

Rescorla, R. A., & Wagner, A. R. (1972). A theory of Pavlovian conditioning: Variations in the effectiveness of reinforcement and non-reinforcement. In A. H. Black & W. F. Prokasy (Eds.), Classical Conditioning II: Current Research and Theory New York: Appleton-Century-Crofts.

Rumelhart, D. E., Hinton, G. E., & Williams, R. J. (1986). Learning internal representations by error propagation. In D. Rumelhart & J. McClelland (Eds.), Parallel Distributed Processing: Explorations in the Microstructure of Cognition (Vol. 1: Foundations) (pp. 318-362). Cambridge, MA: MIT Press.

Schmaltz, L. W., & Theios, J. (1972). Acquisition and extinction of a classically conditioned response in hippocampectomized rabbits (Oryctolagus cuniculus). Journal of Comparative and Physiological Psychology, 79, 328-333 .

Scoville, W. B., & Milner, B. (1957). Loss of recent memory after bilateral hippocampal lesions. Journal of Neurology, Neurosurgery, & Psychiatry, 20, 11-21.

Shepard, R. N. (1958). Stimulus and response generalization: Deduction of the generalization gradient from a trace model. Psychological Review, 65, 242-256.

Solomon, P. R., & Moore, J. W. (1975). Latent inhibition and stimulus generalization of the classically conditioned nictitating membrane response in rabbits (Oryctolagus cuniculus) following dorsal hippocampal ablation. Journal of Comparative and Physiological Psychology, 89, 1192-1203.

Solomon, P., Solomon, S., van der Schaaf, E. & Perry, H. (1983). Altered activity in the hippocampus is more detrimental to classical conditioning than removing the structure. Science, 220, 329-331.

Squire, L. R. (1987). Memory and brain. New York: Oxford University Press.

Sutherland, N. & Mackintosh, N. (1971). Mechanisms of Animal Discrimination Learning. New York: Academic Press.

Sutherland, R. J., & Rudy, J. W. (1989). Configural association theory: The role of the hippocampal formation in learning, memory, and amnesia. Psychobiology, 17 (2), 129-144.

Widrow, B., & Hoff, M. (1960). Adaptive switching circuits. Institute of Radio Engineers, Western Electronic Show and Convention, Convention Record, 4, 96-194.

Winocur, G., Rawlins, J. & Gray, J. R. (1987). The hippocampus and conditioning to contextual cues. Behavioral Neuroscience, 101, 617-625.
